# Gates

**Tom Minka**
Microsoft Research Ltd.
Cambridge, UK

**John Winn**
Microsoft Research Ltd.
Cambridge, UK

## Abstract

Gates are a new notation for representing mixture models and context-sensitive independence in factor graphs. Factor graphs provide a natural representation for message-passing algorithms, such as expectation propagation. However, message passing in mixture models is not well captured by factor graphs unless the entire mixture is represented by one factor, because the message equations have a containment structure. Gates capture this containment structure graphically, allowing both the independences and the message-passing equations for a model to be readily visualized. Different variational approximations for mixture models can be understood as different ways of drawing the gates in a model. We present general equations for expectation propagation and variational message passing in the presence of gates.

## 1   Introduction

Graphical models, such as Bayesian networks and factor graphs [1], are widely used to represent and visualise fixed dependency relationships between random variables. Graphical models are also commonly used as data structures for inference algorithms since they allow independencies between variables to be exploited, leading to significant efficiency gains. However, there is no widely used notation for representing *context-specific* dependencies, that is, dependencies which are present or absent conditioned on the state of another variable in the graph [2]. Such a notation would be necessary not only to represent and communicate context-specific dependencies, but also to be able to exploit context-specific independence to achieve efficient and accurate inference.

A number of notations have been proposed for representing context-specific dependencies, including: case factor diagrams [3], contingent Bayesian networks [4] and labeled graphs [5]. None of these has been widely adopted, raising the question: what properties would a notation need, to achieve widespread use? We believe it would need to be:

- simple to understand and use,
- flexible enough to represent context-specific independencies in real world problems,
- usable as a data structure to allow existing inference algorithms to exploit context-specific independencies for efficiency and accuracy gains,
- usable in conjunction with existing representations, such as factor graphs.

This paper introduces the *gate*, a graphical notation for representing context-specific dependencies that we believe achieves these desiderata. Section 2 describes what a gate is and shows how it can be used to represent context-specific independencies in a number of example models. Section 3 motivates the use of gates for inference and section 4 expands on this by showing how gates can be used within three standard inference algorithms: Expectation Propagation (EP), Variational Message Passing (VMP) and Gibbs sampling. Section 5 shows how the placement of gates can tradeoff cost versus accuracy of inference. Section 6 discusses the use of gates to implement inference algorithms.

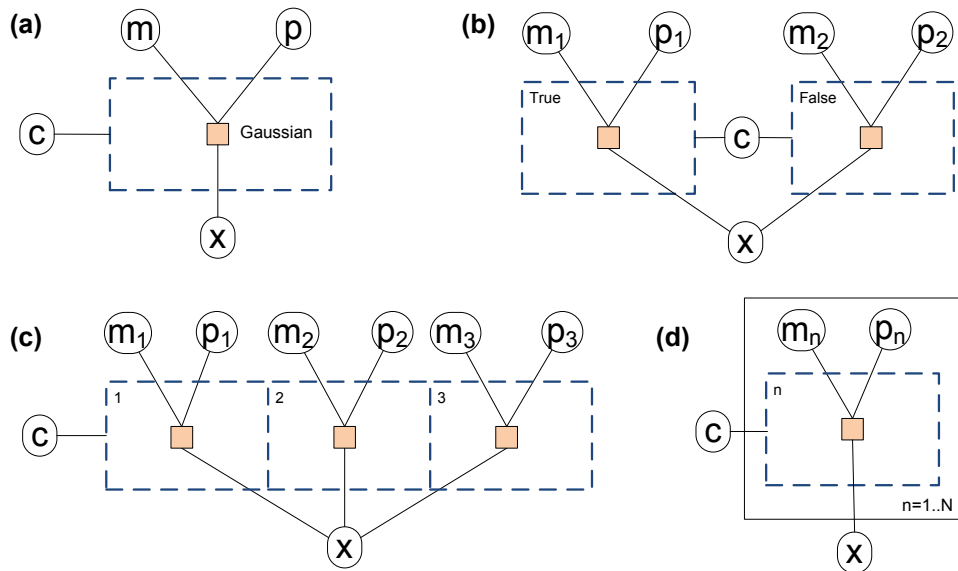

Figure 1: **Gate examples** (a) The dashed rectangle indicates a gate containing a Gaussian factor, with selector variable $c$. (b) Two gates with different key values used to construct a mixture of two Gaussians. (c) When multiple gates share a selector variable, they can be drawn touching with the selector variable connected to only one of the gates. (d) A mixture of $N$ Gaussians constructed using both a gate and a plate. For clarity, factors corresponding to variable priors have been omitted.

## 2   The Gate

A gate encloses part of a factor graph and switches it on or off depending on the state of a latent selector variable. The gate is on when the selector variable has a particular value, called the *key*, and off for all other values. A gate allows context-specific independencies to be made explicit in the graphical model: the dependencies represented by any factors inside the gate are present only in the context of the selector variable having the key value. Mathematically, a gate represents raising the contained factors to the power zero if the gate is off, or one if it is on: $\left(\prod_i f_i(x)\right)^{\delta(c=key)}$ where $c$ is the selector variable. In diagrams, a gate is denoted by a dashed box labelled with the value of *key*, with the selector variable connected to the box boundary. The label may be omitted if $c$ is boolean and *key* is *true*. Whilst the examples in this paper refer to factor graphs, gate notation can also be used in both directed Bayesian networks and undirected graphs.

A simple example of a gate is shown in figure 1a. This example represents the term $\mathcal{N}(x; m, p^{-1})^{\delta(c=true)}$ so that when $c$ is true the gate is on and $x$ has a Gaussian distribution with mean $m$ and precision $p$. Otherwise, the gate is off and $x$ is uniformly distributed (since it is connected to nothing).

By using several gates with different key values, multiple components of a mixture can be represented. Figure 1b shows how a mixture of two Gaussians can be represented using two gates with different key values, true and false. If $c$ is true, $x$ will have distribution $\mathcal{N}(m_1, p_1^{-1})$, otherwise $x$ will have distribution $\mathcal{N}(m_2, p_2^{-1})$. When multiple gates have the same selector variable but different key values, they can be drawn as in figure 1c, with the gate rectangles touching and the selector variable connected to only one of the gates. Notice that in this example, an integer selector variable is used and the key values are the integers 1,2,3.

For large homogeneous mixtures, gates can be used in conjunction with plates [6]. For example, figure 1d shows how a mixture of $N$ Gaussians can be represented by placing the gate, Gaussian factor and mean/precision variables inside a plate, so that they are replicated $N$ times.

Gates may be nested inside each other, implying a conjunction of their conditions. To avoid ambiguities, gates cannot partially overlap, nor can a gate contain its own selector variable.

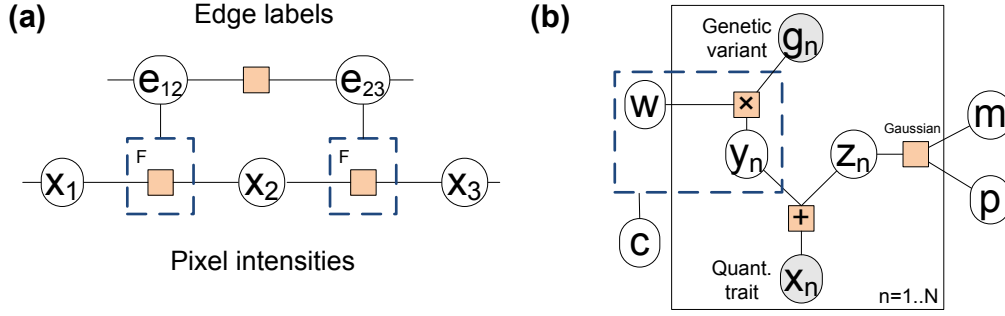

Figure 2: **Examples of models which use gates** (a) A line process where neighboring pixel intensities are independent if an edge exists between them. (b) Testing for dependence between a genetic variant $g_n$ and an observed quantitative trait $x_n$. The selector variable $c$ encodes whether the linear dependency represented by the structure inside the gate is present or absent.

Gates can also contain variables, as well as factors. Such variables have the behaviour that, when the gate is off, they revert to having a default value of false or zero, depending on the variable type. Mathematically, a variable inside a gate represents a Dirac delta when the gate is off: $\delta(x)^{1-\delta(c=key)}$ where $\delta(x)$ is one only when $x$ has its default value. Figure 2b shows an example where variables are contained in gates – this example is described in the following section.

## 2.1   Examples of models with gates

Figure 2a shows a *line process* from [7]. The use of gates makes clear the assumption that two neighboring image pixels $x_i$ and $x_j$ have a dependency between their intensity values, unless there is an edge $e_{ij}$ between them. An opaque three-way factor would hide this context-specific independence.

Gates can also be used to test for independence. In this case the selector variable is connected only to the gate, as shown in the example of figure 2b. This is a model used in functional genomics [8] where the aim is to detect associations between a genetic variant $g_n$ and some quantitative trait $x_n$ (such as height, weight, intelligence etc.) given data from a set of $N$ individuals. The binary selector variable $c$ switches on or off a linear model of the genetic variant's contribution $y_n$ to the trait $x_n$, across all individuals. When the gate is off, $y_n$ reverts to the default value of $0$ and so the trait is explained only by a Gaussian-distributed background model $z_n$. Inferring the posterior distribution of $c$ allows associations between the genetic variation and the trait to be detected.

## 3   How gates arise from message-passing on mixture models

Factor graph notation arises naturally when describing message passing algorithms, such as the sum-product algorithm. Similarly, the gate notation arises naturally when considering the behavior of message passing algorithms on mixture models.

As a motivating example, consider the mixture model of figure 1b when the precisions $p_1$ and $p_2$ are constant. Using 1 and 2 as keys instead of *true* and *false*, the joint distribution is: $p(x, c, m_1, m_2) = p(c)p(m_1)p(m_2)f(x|m_1)^{\delta(c-1)}f(x|m_2)^{\delta(c-2)}$ where $f$ is the Gaussian distribution. If we apply mean-field approximation to this model, we obtain the following fixed-point system:

$$q(c = k) \propto p(c = k) \exp\left(\sum_x q(x) \sum_{m_k} q(m_k) \log f(x|m_k)\right) \tag{1}$$

$$q(m_k) \propto p(m_k) \exp\left(\sum_x q(x) \log f(x|m_k)\right)^{q(c=k)} \tag{2}$$

$$q(x) \propto \prod_k \exp\left(\sum_{m_k} q(m_k) \log f(x|m_k)\right)^{q(c=k)} \tag{3}$$

These updates can be interpreted as message-passing combined with "blurring" (raising to a power between 0 and 1). For example, the update for $q(m_k)$ can be interpreted as (message from prior)×(blurred message from $f$). The update for $q(x)$ can be interpreted as (blurred message from $m_1$)×(blurred message from $m_2$). Blurring occurs whenever a message is sent from a factor having a random exponent to a factor without that exponent. Thus the exponent acts like a container, affecting all messages that pass out of it. Hence, we use a graphical notation where a gate is a container, holding all the factors switched by the gate. Graphically, the blurring operation then happens whenever a message leaves a gate. Messages passed into a gate and within a gate are unchanged.

This graphical property holds true for other algorithms as well. For example, EP on this model will blur the message from $f$ to $m_k$ and from $f$ to $x$, where "blurring" means a linear combination with the 1 function followed by KL-projection.

## 3.1 Why gates are not equivalent to 'pick' factors

It is possible to rewrite this model so that the $f$ factors do not have exponents, and therefore would not be in gates. However, this will necessarily change the approximation. This is because the blurring effect caused by exponents operates in one direction only, while the blurring effect caused by intermediate factors is always bidirectional. For example, suppose we try to write the model using a factor $\mathtt{pick}(x|c, h_1, h_2) = \delta(x - h_1)^{\delta(c-1)}\delta(x - h_2)^{\delta(c-2)}$. We can introduce latent variables $(h_1, h_2)$ so that the model becomes $p(x, c, m_1, m_2, h_1, h_2) = p(c)p(m_1)p(m_2)f(h_1|m_1)f(h_2|m_2)\mathtt{pick}(x|c, h_1, h_2)$. The $\mathtt{pick}$ factor will correctly blur the downward messages from $(m_1, m_2)$ to $x$. However, the $\mathtt{pick}$ factor will also blur the message upward from $x$ before it reaches the factor $f$, which is incorrect.

Another approach is to pick from $(m_1, m_2)$ before reaching the factor $f$, so that the model becomes $p(x, c, m_1, m_2, m) = p(c)p(m_1)p(m_2)f(x|m)\mathtt{pick}(m|c, m_1, m_2)$. In this case, the message from $x$ to $f$ is not blurred, and the upward messages to $(m_1, m_2)$ are blurred, which is correct. However, the downward messages from $(m_1, m_2)$ to $f$ are blurred before reaching $f$, which is incorrect.

## 3.2 Variables inside gates

Now consider an example where it is natural to consider a variable to be inside a gate. The model is: $p(x, c, m_1, m_2, y) = p(c)p(m_1)p(m_2)\prod_k (f_1(x|y)f_2(y|m_k))^{\delta(c-k)}$. If we use a structured variational approximation where $y$ is conditioned on $c$, then the fixed-point equations are [9]:

$$q(c = k) \propto p(c = k) \exp\left(\sum_x q(x) \sum_y q(y|c = k) \log f_1(x|y)\right)$$

$$\exp\left(\sum_y q(y|c = k) \sum_{m_k} q(m_k) \log f_2(y|m_k)\right) \exp\left(-\sum_y q(y|c = k) \log q(y|c = k)\right)$$

$$\tag{4}$$

$$q(y|c = k) \propto \exp\left(\sum_x q(x) \log f_1(x|y)\right) \exp\left(\sum_{m_k} q(m_k) \log f_2(y|m_k)\right) \tag{5}$$

$$q(m_k) \propto p(m_k) \exp\left(\sum_y q(y|c = k) \log f_2(y|m_k)\right)^{q(c=k)} \tag{6}$$

$$q(x) \propto \prod_k \exp\left(\sum_y q(y|c = k) \log f_1(x|y)\right)^{q(c=k)} \tag{7}$$

Notice that only the messages to $x$ and $m_k$ are blurred; the messages to and from $y$ are not blurred. Thus we can think of $y$ as sitting inside the gate. The message from the gate to $c$ can be interpreted as the evidence for the submodel containing $f_1$, $f_2$, and $y$.

# 4 Inference with gates

In the previous section, we explained why the gate notation arises when performing message passing in some example mixture models. In this section, we describe how gate notation can be generally incorporated into Variational Message Passing [10], Expectation Propagation [11] and Gibbs Sampling [7] to allow each of these algorithms to support context-specific independence.

For reference, Table 1 shows the messages needed to apply standard EP or VMP using a fully factorized approximation $q(\mathbf{x}) = \prod_i q(x_i)$. Notice that VMP uses different messages to and from deterministic factors, that is, factors which have the form $f_a(x_i, \mathbf{x}_{a\setminus i}) = \delta(x_i - h(\mathbf{x}_{a\setminus i}))$ where $x_i$ is the derived child variable. Different VMP messages are also used to and from such deterministic derived variables. For both algorithms the marginal distributions are obtained as $q(x_i) = \prod_a m_{a\to i}(x_i)$, except for derived child variables in VMP where $q(x_i) = m_{\text{par}\to i}(x_i)$. The (approximate) model evidence is obtained by a product of contributions, one from each variable and each factor. Table 1 shows these contributions for each algorithm, with the exception that deterministic factors and their derived variables contribute 1 under VMP.

When performing inference on models with gates, it is useful to employ a *normalised form* of gate model. In this form, variables inside a gate have no links to factors outside the gate, and a variable outside a gate links to at most one factor inside the gate. Both of these requirements can be achieved by splitting a variable into a copy inside and a copy outside the gate, connected by an equality factor inside the gate. A factor inside a gate should not connect to the selector of the gate; it should be given the key value instead. In addition, gates should be *balanced* by ensuring that if a variable links

| Alg. | Type | Variable to factor $m_{i\to a}(x_i)$ | Factor to variable $m_{a\to i}(x_i)$ |
|---|---|---|---|
| EP | Any | $\displaystyle\prod_{b\neq a} m_{b\to i}(x_i)$ | $\dfrac{\text{proj}\left[\sum_{\mathbf{x}_a\setminus x_i}\left(\prod_{j\in a} m_{j\to a}(x_j)\right) f_a(\mathbf{x}_a)\right]}{m_{i\to a}(x_i)}$ |
| VMP | Stochastic | $\displaystyle\prod_{a\ni i} m_{a\to i}(x_i)$ | $\exp\left[\sum_{\mathbf{x}_a\setminus x_i}\left(\prod_{j\neq i} m_{j\to a}(x_j)\right)\log f_a(\mathbf{x}_a)\right]$ |
| | Det. to parent | $\displaystyle\prod_{b\neq a} m_{b\to i}(x_i)$ | $\exp\left[\sum_{\mathbf{x}_{a\setminus(i,\text{ch})}}\left(\prod_{k\neq(i,\text{ch})} m_{k\to a}(x_k)\right)\log \hat{f}_a(\mathbf{x}_a)\right]$ where $\hat{f}_a(\mathbf{x}_a) = \sum_{x_{\text{ch}}} m_{\text{ch}\to a}(x_{\text{ch}}) f_a(\mathbf{x}_a)$ |
| | Det. to child | $m_{\text{par}\to i}(x_i)$ | $\text{proj}\left[\sum_{\mathbf{x}_a\setminus x_i}\left(\prod_{j\neq i} m_{j\to a}(x_j)\right) f_a(\mathbf{x}_a)\right]$ |

| Alg. | Evidence for variable $x_i$ | Evidence for factor $f_a$ |
|---|---|---|
| EP | $s_i = \sum_{x_i}\prod_a m_{a\to i}(x_i)$ | $s_a = \dfrac{\sum_{\mathbf{x}_a}\left(\prod_{j\in a} m_{j\to a}(x_j)\right) f_a(\mathbf{x}_a)}{\sum_{\mathbf{x}_a}\prod_{j\in a} m_{j\to a}(x_j) m_{a\to j}(x_j)}$ |
| VMP | $s_i = \exp(-\sum_{x_i} q(x_i)\log q(x_i))$ | $s_a = \exp\left(\sum_{\mathbf{x}_a}\left(\prod_{j\in a} m_{j\to a}(x_j)\right)\log f_a(\mathbf{x}_a)\right)$ |

Table 1: **Messages and evidence computations for EP and VMP** The top part of the table shows messages between a variable $x_i$ and a factor $f_a$. The notation $j \in a$ refers to all neighbors of the factor, $j \neq i$ is all neighbors except $i$, *par* is the parent factor of a derived variable, and *ch* is the child variable of a deterministic factor. The $\text{proj}[p]$ operator returns an exponential-family distribution whose sufficient statistics match $p$. The bottom part of the table shows the evidence contributions for variables and factors in each algorithm.

to a factor in a gate with selector variable $c$, the variable also links to factors in gates keyed on all other values of the selector variable $c$. This can be achieved by connecting the variable to uniform factors in gates for any missing values of $c$. After balancing, each gate is part of a *gate block* – a set of gates activated by different values of the same condition variable. See [12] for details.

## 4.1 Variational Message Passing with gates

VMP can be augmented to run on a gate model in normalised form, by changing only the messages out of the gate and by introducing messages from the gate to the selector variable. Messages sent between nodes inside the gate and messages into the gate are unchanged from standard VMP. The variational distributions for variables inside gates are implicitly conditioned on the gate selector, as at the end of section 3. In the following, an individual gate is denoted $g$, its selector variable $c$ and its key $k_g$. See [12] for the derivations.

The messages out of a gate are modified as follows:

- The message from a factor $f_a$ inside a gate $g$ with selector $c$ to a variable outside $g$ is the usual VMP message, raised to the power $m_{c \to g}(c = k_g)$, except in the following case.
- Where a variable $x_i$ is the child of a number of deterministic factors inside a gate block $G$ with selector variable $c$, the variable is treated as derived and the message is a moment-matched average of the individual VMP messages. Then the message to $x_i$ is

$$m_{G \to i}(x_i) = \text{proj}\left[\sum_{g \in G} m_{c \to g}(c = k_g) m_{g \to i}(x_i)\right] \tag{8}$$

  where $m_{g \to i}(x_i)$ is the usual VMP message from the unique parent factor in $g$ and proj is a moment-matching projection onto the exponential family.

The message from a gate $g$ to its selector variable $c$ is a product of evidence messages from the contained nodes:

$$m_{g \to c}(c = k_g) = \prod_{a \in g} s_a \prod_{i \in g} s_i, \qquad m_{g \to c}(c \neq k_g) = 1 \tag{9}$$

where $s_a$ and $s_i$ are the VMP evidence messages from a factor and variable, respectively (Table 1). The set of contained factors includes any contained gates, which are treated as single factors by the containing gate. Deterministic variables and factors send evidence messages of 1, except where a deterministic factor $f_a$ parents a variable $x_i$ outside $g$. Instead of sending $s_a = 1$, the factor sends:

$$s_a = \exp\left(\sum_{x_i} m_{a \to i}(x_i) \log m_{i \to a}(x_i)\right) \tag{10}$$

The child variable $x_i$ outside the gate also has a different evidence message:

$$s_i = \exp\left(-\sum_{x_i} m_{G \to i}(x_i) \log m_{i \to a}(x_i)\right) \tag{11}$$

where $m_{G \to i}$ is the message from the parents (8) and $m_{i \to a}$ is the message from $x_i$ to any parent. To allow for nested gates, we must also define an evidence message for a gate:

$$s_g = \left(\prod_{a \in g} s_a \prod_{i \in g} s_i\right)^{q(c = k_g)} \tag{12}$$

## 4.2 Expectation Propagation with gates

As with VMP, EP can support gate models in normalised form by making small modifications to the message-passing rules. Once again, messages between nodes inside a gate are unchanged. Recall that, following gate balancing, all gates are part of gate blocks. In the following, an individual gate is denoted $g$, its selector variable $c$ and its key $k_g$. See [12] for the derivations.

The messages into a gate are as follows:

- The message from a selector variable to each gate in a gate block $G$ is the same. It is the product of all messages into the variable excluding messages from gates in $G$.
- The message from a variable to each neighboring factor inside a gate block $G$ is the same. It is product of all messages into the variable excluding messages from any factor in $G$.

Let $\mathrm{nbrs}(g)$ be the set of variables outside of $g$ connected to some factor in $g$. Each gate computes an intermediate evidence-like quantity $s_g$ defined as:

$$s_g = \prod_{a \in g} s_a \prod_{i \in g} s_i \prod_{i \in \mathrm{nbrs}(g)} s_{ig} \qquad \text{where } s_{ig} = \sum_{x_i} m_{i \to g}(x_i) m_{g \to i}(x_i) \tag{13}$$

where $m_{g \to i}$ is the usual EP message to $x_i$ from its (unique) neighboring factor in $g$. The third term is used to cancel the denominators of $s_a$ (see definition in Table 1). Given this quantity, the messages out of a gate may now be specified:

- The combined message from all factors in a gate block $G$ with selector variable $c$ to a variable $x_i$ is the weighted average of the messages sent by each factor:

$$m_{G \to i}(x_i) = \frac{\mathrm{proj}\left[\sum_{g \in G} m_{c \to g}(c = k_g) s_g s_{ig}^{-1} m_{g \to i}(x_i) m_{i \to g}(x_i)\right]}{m_{i \to g}(x_i)} \tag{14}$$

  (Note $m_{i \to g}(x_i)$ is the same for each gate $g$.)
- The message from a gate block $G$ to its selector variable $c$ is:

$$m_{G \to c}(c = k_g) = \frac{s_g}{\sum_{g \in G} s_g} \tag{15}$$

Finally, the evidence contribution of a gate block with selector $c$ is:

$$s_c = \frac{\sum_{g \in G} s_g}{\prod_{i \in \mathrm{nbrs}(g)} \sum_{x_i} m_{i \to g}(x_i) m_{G \to i}(x_i)} \tag{16}$$

### 4.3 Gibbs sampling with gates

Gibbs sampling can easily extend to gates which contain only factors. Gates containing variables require a facility for computing the evidence of a submodel, which Gibbs sampling does not provide. Note also that Gibbs sampling does not support deterministic factors. Thus the graph should only be normalised up to these constraints. The algorithm starts by setting the variables to initial values and sending these values to their neighboring factors. Then for each variable $x_i$ in turn:

1. Query each neighboring factor for a conditional distribution for $x_i$. If the factor is in a gate that is currently off, replace with a uniform distribution. For a gate $g$ with selector $x_i$, the conditional distribution is proportional to $s$ for the key value and $1$ otherwise, where $s$ is the product of all factors in $g$.
2. Multiply the distributions from neighboring factors together to get the variable's conditional distribution. Sample a new value for the variable from its conditional distribution.

## 5 Enlarging gates to increase approximation accuracy

Gates induce a structured approximation as in [9], so by moving nodes inside or outside of gates, you can trade off inference accuracy versus cost. Because one gate of a gate block is always on, any node (variable or factor) outside a gate block $G$ can be equivalently placed inside each gate of $G$. This increases accuracy since a separate set of messages will be maintained for each case, but it may increase the cost.

For example, Archambeau and Verleysen [14] suggested a structured approximation for Student-t mixture models, instead of the factorised approximation of [13]. Their modification can be viewed as a gate enlargement (figure 3). By enlarging the gate block to include $u_{nm}$, the blurring between the multiplication factor and $u_{nm}$ is removed, increasing accuracy. This comes at no additional cost since $u_{nm}$ is only used by one gate and therefore only one message is needed per $n$ and $m$.

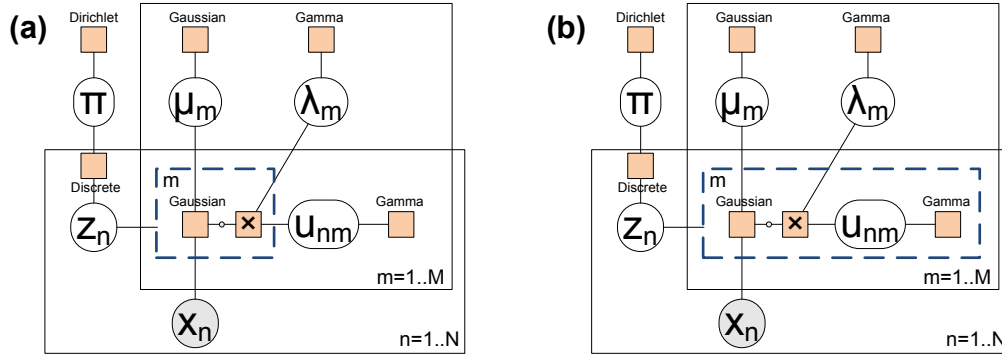

Figure 3: **Student-t mixture model using gates** (a) Model from [13] (b) Structured approximation suggested by [14], which can be interpreted as enlarging the gate.

## 6   Discussion and conclusions

Gates have proven very useful to us when implementing a library for inference in graphical models. By using gates, the library allows mixtures of arbitrary sub-models, such as mixtures of factor analysers. Gates are also used for computing the evidence for a model, by placing the entire model in a gate with binary selector variable $b$. The log evidence is then the log-odds of $b$, that is, $\log P(b = \text{true}) - \log P(b = \text{false})$. Similarly, gates are used for model comparison by placing each model in a different gate of a gate block. The marginal over the selector gives the posterior distribution over models.

Graphical models not only provide a visual way to represent a probabilistic model, but they can also be used as a data structure for performing inference on that model. We have shown that gates are similarly effective both as a graphical modelling notation and as a construct within an inference algorithm.

## References

[1] B. Frey, F. Kschischang, H. Loeliger, and N. Wiberg. Factor graphs and algorithms. In *Proc. of the 35th Allerton Conference on Communication, Control and Computing*, 1998.

[2] C. Boutilier, N. Friedman, M. Goldszmidt, and D. Koller. Context-specific independence in Bayesian networks. In *Proc. of the 12th conference on Uncertainty in Artificial Intelligence*, pages 115–123, 1996.

[3] D. McAllester, M. Collins, and F. Pereira. Case-factor diagrams for structured probabilistic modeling. *Uncertainty in Artificial Intelligence*, 2004.

[4] B. Milch, B. Marthi, D. Sontag, S. Russell, D. L. Ong, and A. Kolobov. Approximate inference for infinite contingent Bayesian networks. In *Proc. of the 6th workshop on Artificial Intelligence and Statistics*, 2005.

[5] E. Mjolsness. Labeled graph notations for graphical models: Extended report. Technical Report TR# 04-03, UCI ICS, March 2004.

[6] W. L. Buntine. Operations for learning with graphical models. *JAIR*, 2:159–225, 1994.

[7] S. Geman and D. Geman. Stochastic relaxation, Gibbs distribution, and the Bayesian restoration of images. *IEEE Trans. on Pattern Anal. Machine Intell.*, 6:721–741, 1984.

[8] E. S. Lander and D. Botstein. Mapping Mendelian factors underlying quantitative traits using RFLP linkage maps. *Genetics*, 121(1):185–199, 1989.

[9] W.A.J.J. Wiegerinck. Variational approximations between mean field theory and the junction tree algorithm. In *UAI*, pages 626–633, 2000.

[10] J. Winn and C. M. Bishop. Variational Message Passing. *JMLR*, 6:661–694, 2005.

[11] T. P. Minka. Expectation propagation for approximate Bayesian inference. In *UAI*, pages 362–369, 2001.

[12] T. Minka and J. Winn. Gates: A graphical notation for mixture models. Technical report, Microsoft Research Ltd, 2008.

[13] M. Svensén and C. M. Bishop. Robust Bayesian mixture modelling. *Neurocomputing*, 64:235–252, 2005.

[14] C. Archambeau and M. Verleysen. Robust Bayesian clustering. *Neural Networks*, 20:129–138, 2007.

